# Relaxed Clipping: A Global Training Method for Robust Regression and Classification

**Yaoliang Yu, Min Yang, Linli Xu, Martha White, Dale Schuurmans**
University of Alberta, Dept. Computing Science, Edmonton AB T6G 2E8, Canada
{yaoliang,myang2,linli,whitem,dale}@cs.ualberta.ca

## Abstract

Robust regression and classification are often thought to require non-convex loss functions that prevent scalable, global training. However, such a view neglects the possibility of reformulated training methods that can yield practically solvable alternatives. A natural way to make a loss function more robust to outliers is to truncate loss values that exceed a maximum threshold. We demonstrate that a relaxation of this form of "loss clipping" can be made globally solvable and applicable to any standard loss while guaranteeing robustness against outliers. We present a generic procedure that can be applied to standard loss functions and demonstrate improved robustness in regression and classification problems.

## 1 Introduction

Robust statistics is a well established field that analyzes the sensitivity of common estimators to outliers and provides alternative estimators that achieve improved robustness [11, 13, 17, 23]. Outliers are understood to be observations that have been corrupted, incorrectly measured, mis-recorded, drawn under different conditions than those intended, or so atypical as to require separate modeling. The main goal of classical robust statistics is to make estimators invariant, or nearly invariant, to *arbitrary* changes made to a non-trivial fraction of the sample data—a goal that is equally relevant to machine learning research given that data sets are often collected with limited or no quality control, making outliers ubiquitous. Unfortunately, the state-of-the-art in robust statistics relies on non-convex training criteria that have yet to yield efficient global solution methods [13, 17, 23].

Although many robust regression methods have been proposed in the classical literature, *M-estimators* continue to be a dominant approach [13, 17]. These correspond to the standard machine learning approach of minimizing a sum of prediction errors under a given loss function (assuming a fixed scaling). M-estimation is reasonably well understood, analytically tractable, and provides a simple framework for trading off between robustness against outliers and data efficiency on inliers [13, 17]. Unfortunately, robustness in this context comes with a cost: when minimizing a convex loss, even a single data point can dominate the result. That is, *any (non-constant) convex loss function exhibits necessarily* unbounded *sensitivity to even a single outlier* [17, §5.4.1]. Although unbounded sensitivity can obviously be mitigated by imposing prior bounds on the domain and range of the data [5, 6], such is not always possible in practice. Instead, the classical literature achieves bounded outlier sensitivity by considering *redescending* loss functions (see [17, §2.2] for a definition), or more restrictively, *bounded* loss functions, both of which are inherently non-convex. Robust regression has also been extensively investigated in computer vision [2, 26], where a similar conclusion has been reached that bounded loss functions are necessary to counteract the types of outliers created by edge discontinuities, multiple motions, and specularities in image data.

For *classification* the story is similar. The attempt to avoid outlier sensitivity has led many to propose bounded loss functions [8, 15, 18, 19, 25] to replace the standard convex, unbounded losses deployed in support vector machines and boosting [9] respectively. In fact, [16] has shown that minimizing

*any* convex margin loss cannot achieve robustness to random misclassification noise. The conclusion reached in the classification literature, as in the regression literature, is therefore that non-convexity is *necessary* to ensure robustness against outliers—creating an apparent dilemma: one can achieve global training via convexity or outlier robustness via boundedness, but not both.

In this paper we present a counterpoint to these pessimistic conclusions. In particular, we present a general model for bounding any convex loss function, via a process of "loss clipping", that ensures bounded sensitivity to outliers. Although the resulting optimization problem is not, by itself, convex, we demonstrate an efficient convex relaxation and rounding procedure that guarantees bounded response to data—a guarantee that cannot be established for any convex loss minimization on its own. The approach we propose is generic and can be applied to any standard loss function, be it for regression or classification. Our work is inspired by a number of studies that have investigated robust estimators in computer vision and machine learning [2, 26, 27, 30]. However, these previous attempts were either hampered by local optimization or restricted to special cases; none had guarantees of global training and outlier insensitivity.

Before proceeding it is important to realize that there are many alternative conceptions of "robustness" in the literature that do not correspond to the notion we are investigating. For example, work on "robust optimization" [28, 29] considers minimizing the worst case loss achieved given prespecified bounds on the maximum data deviation that will be considered. Although interesting, these results do not directly bear on the question at hand since we explicitly do not bound the magnitude of the outliers (i.e. the degree of leverage [23, §1.1], nor the size of response deviations). Another notion of robustness is algorithmic stability under leave-one-out perturbation [3]. Although loosely related, algorithmic stability addresses the analysis of given learning procedures rather than describing how a stable algorithm might be generally achieved in the presence of arbitrary outliers. We also do not focus on asymptotic or infinitesimal notions from robust statistics, such as influence functions [11], nor impose boundedness assumptions on the domain and range of the data or the predictor [5, 6].

## 2    Background

We consider the standard supervised setting where one is given an input matrix $X$ and output targets $\mathbf{y}$, with the goal of learning a predictor $h : \Re^m \to \Re$. Each row of $X$ gives the feature representation for one training example, denoted $X_{i:}$, with corresponding target $y_i$. We will assume the predictor can be written as a generalized linear model; that is, the predictions are given by $\hat{y}_i = f(X_{i:}\boldsymbol{\theta})$ for a fixed transfer function $f$ (possibly identity) and a vector of parameters $\boldsymbol{\theta}$. For training, we will consider the standard $L_2$ regularized loss minimization problem

$$\min_{\boldsymbol{\theta}} \frac{\gamma}{2}\|\boldsymbol{\theta}\|_2^2 + \sum_{i=1}^{n} L(y_i, \hat{y}_i) \quad = \quad \min_{\boldsymbol{\theta}} \frac{\gamma}{2}\|\boldsymbol{\theta}\|_2^2 + \sum_{i=1}^{n} L(y_i, f(X_{i:}\boldsymbol{\theta})) \tag{1}$$

where $L$ denotes the loss function, $\gamma$ is the regularization constant, and $n$ denotes the number of training examples. Normally the loss function $L$ is chosen to be convex in $\boldsymbol{\theta}$ so that the minimization problem can be solved efficiently. Although convexity is important for computational tractability, it has the undesired side-effect of causing unbounded outlier sensitivity, as mentioned. Obviously, the severity of the problem will range from minimal to extreme depending on the nature of the distribution over $(\mathbf{x}, y)$. Nevertheless, our goal in this paper will be to eliminate unbounded sensitivity for convex loss functions while retaining a scalable computational approach.[1]

**Standard Convex Loss Functions:**    Our general construction applies to arbitrary convex losses, but we will demonstrate our methods on standard loss functions employed in regression and classification. A standard example is *Bregman divergences*, which are defined by taking a strongly convex differentiable potential $\Phi$ then taking the difference between the potential and its first order Taylor approximation, obtaining a loss $L_\Phi(\hat{y}\|y) = \Phi(\hat{y}) - \Phi(y) - \phi(y)(\hat{y} - y)$, where $\phi(y) = \Phi'(y)$ [1, 14]. Several natural loss functions can be defined this way, including least squares $L_\Phi(\hat{y}\|y) = (\hat{y} - y)^2/2$, using the potential $\Phi(y) = y^2/2$, and forward KL-divergence $L_\Phi(\hat{y}\|y) = \hat{y}\ln\frac{\hat{y}}{y} + (1 - \hat{y})\ln\frac{1-\hat{y}}{1-y}$, using the potential $\Phi(y) = y\ln y + (1 - y)\ln(1 - y)$ for $0 \le y \le 1$.

A related construction is *matching losses* [14], which are determined by taking a strictly increasing differentiable transfer function $f$ to be used in prediction via $\hat{y} = f(z)$ where $z = \mathbf{x}^\top \boldsymbol{\theta}$. Then, given a transfer $f$, a loss can be defined by $L_F(\hat{z}\|z) = \int_z^{\hat{z}} f(\zeta) - f(z) \, d\zeta = F(\hat{z}) - F(z) - f(z)(\hat{z} - z)$ such that $F$ satisfies $F'(z) = f(z)$. By definition, matching losses are also Bregman divergences, since $F$ is differentiable and the assumptions on $f$ imply that $F$ is strongly convex. These two loss constructions are related by the equality $L_\Phi(y\|\hat{y}) = L_F(\hat{z}\|z)$ where $F$ is the Legendre-Fenchel conjugate of $\Phi$ [4, §3.3], $z = f^{-1}(y) = \phi(y)$ and $\hat{z} = f^{-1}(\hat{y}) = \phi(\hat{y})$ [1, 14]. For example, the post-prediction KL-divergence $y \ln \frac{y}{\hat{y}} + (1 - y) \ln \frac{1-y}{1-\hat{y}}$ is equal to the convex pre-prediction loss $L_F(\hat{z}\|z) = \ln(e^{\hat{z}} + 1) - \ln(e^z + 1) - \sigma(z)(\hat{z} - z)$ via the transfer $\hat{y} = \sigma(\hat{z}) = (1 + e^{-\hat{z}})^{-1}$. Such losses are prevalent in regression and probabilistic classification settings.

For discrete classification it is also natural to work with a continuous pre-prediction space $\hat{z} = \mathbf{x}^\top \boldsymbol{\theta}$, recovering discrete post-predictions $\hat{y} \in \{-1, 1\}$ via a step transfer $\hat{y} = \text{sign}(z)$. Although a step transfer does not admit the matching loss construction, a surrogate *margin loss* can be obtained by taking a nonincreasing function $l$ such that $\lim_{m \to \infty} l(m) = 0$, then defining $L_l(\hat{y}, y) = l(y\hat{y})$. Here $y\hat{y}$ is known as the *classification margin*. Standard examples include misclassification loss, $L_l(\hat{y}, y) = 1_{(y\hat{y} < 0)}$, support vector machine (hinge) loss, $L_l(\hat{y}, y) = \max(0, 1 - y\hat{y})$, binomial deviance loss, $L_l(\hat{y}, y) = \ln(1 + e^{-y\hat{y}})$ [12], and Adaboost loss, $L_l(\hat{y}, y) = e^{-y\hat{y}}$ [9]. If the margin loss is furthermore chosen to be convex, efficient minimization can be attained.

To unify our presentation below we will simply denote all loss functions by $\ell(y, \mathbf{x}^\top \boldsymbol{\theta})$, with the understanding that $\ell(y, \mathbf{x}^\top \boldsymbol{\theta}) = L_\Phi(\mathbf{x}^\top \boldsymbol{\theta}\|y)$ if the loss is Bregman divergence on potential $\Phi$; $\ell(y, \mathbf{x}^\top \boldsymbol{\theta}) = L_F(\mathbf{x}^\top \boldsymbol{\theta}\|f^{-1}(y))$ if the loss is a matching loss with transfer $f$; and $\ell(y, \mathbf{x}^\top \boldsymbol{\theta}) = l(y\mathbf{x}^\top \boldsymbol{\theta})$ if the loss is a margin loss with margin function $l$. In each case, the loss is convex in the parameters $\boldsymbol{\theta}$. Note that by their very convexity these losses cannot be robust: all admit unbounded sensitivity to a single outlier (the same is also true for $L_1$ loss when applied to regression).

**Bounded loss functions:** As observed, non-convex loss functions are necessary to bound the effects of outliers [17]. Black and Rangarajan [2] provide a useful catalog of bounded and redescending loss functions for robust regression, of which a representative example is the *Geman and McClure loss* $L(y, \hat{y}) = (\hat{y} - y)^2/(\tau + (\hat{y} - y)^2)$ for $\tau > 0$; see Figure 1. Unfortunately, as Figure 1 makes plain, boundedness implies non-convexity (for any non-constant function). It therefore appears that bounded loss functions achieve robustness at the cost of losing global training guarantees. Our goal is to show that robustness and efficient global training are not mutually exclusive. Despite extensive research on regression and classification, almost no work we are aware of (save perhaps [30] in a limited way) attempts to reconcile robustness to outliers with global training algorithms.

## 3 Loss Clipping

Adapting the ideas of [2, 27, 30], given any convex loss $\ell(y, \mathbf{x}^\top \boldsymbol{\theta})$ define the *clipped loss* as

$$\ell_c(y, \mathbf{x}^\top \boldsymbol{\theta}) \quad = \quad \min(1, \ell(y, \mathbf{x}^\top \boldsymbol{\theta})). \qquad (2)$$

Figure 1 demonstrates loss clipping for some standard loss functions. Given a clipped loss, a robust form of training problem (1) can be written as

$$\min_{\boldsymbol{\theta}} \frac{\gamma}{2} \|\boldsymbol{\theta}\|_2^2 + \sum_{i=1}^n \ell_c(y_i, X_{i:}\boldsymbol{\theta}). \qquad (3)$$

Clearly such a training objective bounds the influence of any one training example on the final result. Unfortunately, the formulation (3) is not computationally convenient because the optimization problem it poses is neither convex nor smooth. To make progress on the computational question we exploit a key observation: for any loss function, its corresponding clipped loss can be indirectly expressed by an auxiliary optimization of a smooth objective (if the original loss function itself was smooth). That is, given a loss $\ell(y, \mathbf{x}^\top \boldsymbol{\theta})$ define the corresponding $\rho$-*relaxed loss* to be

$$\ell_\rho(y, \mathbf{x}^\top \boldsymbol{\theta}) \quad = \quad \rho \ell(y, \mathbf{x}^\top \boldsymbol{\theta}) + 1 - \rho \qquad (4)$$

for $0 \le \rho \le 1$; see Figure 1. This construction is an instance of an outlier process as described in [2] and is motivated by a special case hinge-loss construction originally proposed in [30]. The

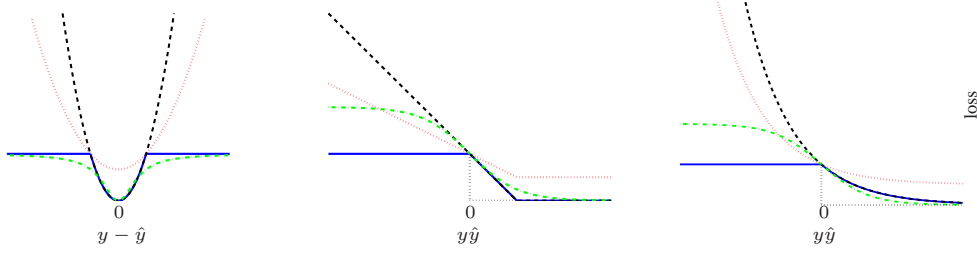

Figure 1: Comparing standard losses (dashed) with corresponding "clipped" losses (solid), $\rho$-relaxed losses (dotted), and non-convex robust losses (dash-dotted). **Left**: squared loss (dashed), clipped (solid), $1/3$-relaxed (dotted), robust Geman and McClure loss [2] (dash-dotted). **Center**: SVM hinge loss (dashed), clipped [27, 30] (solid), $1/2$-relaxed (upper dotted), robust $1 - \tanh(y\hat{y})$ loss [19] (dash-dotted). **Right**: Adaboost exponential loss (dashed), clipped (solid), $1/2$-relaxed (upper dotted), robust $1 - \tanh(y\hat{y})$ loss [19] (dash-dotted).

$\rho$-relaxation provides a convenient characterization of any clipped loss, since it can be shown in general that minimizing a corresponding $\rho$-relaxed loss is equivalent to minimizing the clipped loss.

**Proposition 1** *For any loss function $\ell(y, \mathbf{x}^\top \boldsymbol{\theta})$, we have $\ell_c(y, \mathbf{x}^\top \boldsymbol{\theta}) = \min_{0 \leq \rho \leq 1} \ell_\rho(y, \mathbf{x}^\top \boldsymbol{\theta})$.*

(The proof is straightforward, but it is given in the supplement for completeness.) Proposition 1 now allows us to reformulate (3) as a smooth optimization using the fact that the optimization is completely separable between the $\rho_i$ variables:

$$(3) \quad = \quad \min_{\boldsymbol{\theta}} \min_{\mathbf{0} \leq \boldsymbol{\rho} \leq \mathbf{1}} \frac{\gamma}{2} \|\boldsymbol{\theta}\|_2^2 + \sum_{i=1}^{n} \rho_i \ell(y_i, X_{i:}\boldsymbol{\theta}) + 1 - \rho_i. \tag{5}$$

Unfortunately, the resulting problem is not jointly convex in $\boldsymbol{\rho}$ and $\boldsymbol{\theta}$ even though it is convex in each given the other. Such marginal convexity might suggest that an alternating minimization strategy, however the proof of Proposition 1 shows that each minimization over $\boldsymbol{\rho}$ will result in $\rho_i = 0$ for losses greater than 1, or $\rho_i = 1$ for losses less than 1. Such discrete assignments immediately causes the search to get trapped in local minima, requiring a more sophisticated approach to be considered.

## 4  A Convex Relaxation

One contribution of this paper is to derive an exact reformulation of (5) that admits a convex relaxation and rounding scheme that retain bounded sensitivity to outliers. We first show how the relaxation can be efficiently solved by a scalable algorithm that eliminates any need for semidefinite programming, then provide a guarantee of bounded outlier sensitivity in Section 5.

**Reformulation:**  To ease the notational burden, let us rewrite (5) in matrix-vector form

$$(5) \quad = \quad \min_{\mathbf{0} \leq \boldsymbol{\rho} \leq \mathbf{1}} \min_{\boldsymbol{\theta}} R(\boldsymbol{\rho}, \boldsymbol{\theta}) \tag{6}$$

$$\text{where} \quad R(\boldsymbol{\rho}, \boldsymbol{\theta}) \quad = \quad \frac{\gamma}{2} \|\boldsymbol{\theta}\|^2 + \boldsymbol{\rho}^\top \boldsymbol{\ell}(\mathbf{y}, X\boldsymbol{\theta}) + \mathbf{1}^\top (\mathbf{1} - \boldsymbol{\rho}). \tag{7}$$

Here $\mathbf{1}$ denotes the vector of all 1s, and it is understood that $\boldsymbol{\ell}(\mathbf{y}, X\boldsymbol{\theta})$ refers to the $n \times 1$ vector of individual training losses. Given that $\ell(\cdot, \cdot)$ is convex in its second argument we will be able to exploit Fenchel duality to re-express the min-min form (6) into a min-max form that will serve as the basis for the subsequent relaxation. In particular, consider the definition

$$\ell^*(y, \alpha) \quad = \quad \sup_{\boldsymbol{\theta}} \alpha \mathbf{x}^\top \boldsymbol{\theta} - \ell(y, \mathbf{x}^\top \boldsymbol{\theta}). \tag{8}$$

By construction, $\ell^*(y, \alpha)$ is guaranteed to be convex in $\alpha$ since it is a pointwise maximum over linear functions [4, §3.2].

**Lemma 1** *For any convex differentiable loss function $\ell(y, \mathbf{x}^\top \boldsymbol{\theta})$ such that the level sets of $\ell_\alpha(\mathbf{v}) = \alpha \mathbf{x}^\top (\boldsymbol{\theta} - \mathbf{v}) + \ell(y, \mathbf{x}^\top \mathbf{v})$ are bounded, we have*

$$\ell(y, \mathbf{x}^\top \boldsymbol{\theta}) \quad = \quad \sup_{\alpha} \alpha \mathbf{x}^\top \boldsymbol{\theta} - \ell^*(y, \alpha). \tag{9}$$

(This is a standard result, but a proof is given in the supplement for completeness.) For standard losses $\ell^*(y, \alpha)$ can be computed explicitly [1, 7]. For example, if $\ell(y, \mathbf{x}^\top \boldsymbol{\theta}) = (y - \mathbf{x}^\top \boldsymbol{\theta})^2/2$ then $\ell^*(y, \alpha) = \alpha^2/2 + \alpha y$. Now let $\Delta(\boldsymbol{\alpha})$ denote putting $\boldsymbol{\alpha}$ in the main diagonal of a square matrix and let $\boldsymbol{\ell}^*(\mathbf{y}, \boldsymbol{\alpha})$ refer to the $n \times 1$ vector of dual values over training examples. We can then express the main reformulation as follows.

**Theorem 1** *Let $K = XX^\top$ denote the kernel matrix over input data. Then*

$$(6) \quad = \quad \min_{-\frac{1}{\sqrt{n+1}}\mathbf{1} \leq \boldsymbol{\nu} \leq \frac{1}{\sqrt{n+1}}\mathbf{1},\, \nu_1 = \frac{1}{\sqrt{n+1}},\, \|\boldsymbol{\nu}\|=1} \quad \sup_{\boldsymbol{\alpha}} \quad -(n+1)\,\boldsymbol{\nu}^\top T(\boldsymbol{\alpha})\boldsymbol{\nu} \qquad (10)$$

*where $\boldsymbol{\nu}$ is an $(n+1) \times 1$ variable, $\boldsymbol{\alpha}$ is an $n \times 1$ variable, and the matrix $T(\boldsymbol{\alpha})$ is given by*

$$T(\boldsymbol{\alpha}) = \frac{1}{8\gamma} \begin{bmatrix} \mathbf{1}^\top \\ I \end{bmatrix} \Delta(\boldsymbol{\alpha}) K \Delta(\boldsymbol{\alpha}) \begin{bmatrix} \mathbf{1} & I \end{bmatrix} + \frac{1}{4} \begin{bmatrix} 2(\mathbf{1}^\top \boldsymbol{\ell}^*(\mathbf{y}, \boldsymbol{\alpha}) - n) & (\boldsymbol{\ell}^*(\mathbf{y}, \boldsymbol{\alpha}) + \mathbf{1})^\top \\ \boldsymbol{\ell}^*(\mathbf{y}, \boldsymbol{\alpha}) + \mathbf{1} & 0 \end{bmatrix}. \quad (11)$$

The proof consists in first dualizing $\boldsymbol{\theta}$ in (6) via Lemma 1, which establishes the key relationship

$$\boldsymbol{\theta} = -\frac{1}{\gamma} X^\top \Delta(\boldsymbol{\rho})\boldsymbol{\alpha}. \qquad (12)$$

The remainder of the proof is merely algebra: given a solution $\boldsymbol{\nu}$ to (10), the corresponding solution $\boldsymbol{\rho}$ to (6) can be recovered via $\boldsymbol{\rho} = \frac{1}{2}(\mathbf{1} + \boldsymbol{\nu}_{2:n+1}\sqrt{n+1})$. See the supplement for full details.

Note that the formulation (10) given in Theorem 1 is exact. No approximation to the problem (6) has been introduced to this point. Unfortunately, as in (6), the formulation (10) is still not directly amenable to an efficient algorithm: the objective is concave in $\boldsymbol{\alpha}$, conveniently, but it is not convex in $\boldsymbol{\nu}$. The advantage attained by (10) however is that we can now derive an effective relaxation.

**Relaxation:** Let $\boldsymbol{\delta}(M)$ denote the main diagonal vector of the square matrix $M$ and let $\mathrm{tr}(M)$ denote the trace. Consider the following relaxation

$$(10) \quad \geq \quad \min_{M \succeq 0,\, \boldsymbol{\delta}(M) = \frac{1}{n+1}\mathbf{1}} \quad \sup_{\boldsymbol{\alpha}} \quad -(n+1)\,\mathrm{tr}(MT(\boldsymbol{\alpha})) \qquad (13)$$

$$= \quad \sup_{\boldsymbol{\alpha}} \quad \min_{M \succeq 0,\, \boldsymbol{\delta}(M) = \frac{1}{n+1}\mathbf{1}} \quad -(n+1)\,\mathrm{tr}(MT(\boldsymbol{\alpha})) \qquad (14)$$

where we used strong minimax duality to obtain (14) from (13): since the constraint region on $M$ is compact and the inner objective is concave and convex in $\boldsymbol{\alpha}$ and $M$ respectively, Sion's minimax theorem is applicable [22, §37]. Next enforce the constraint $\boldsymbol{\delta}(M) = \frac{1}{n+1}\mathbf{1}$ with a Lagrange multiplier $\boldsymbol{\lambda}$:

$$(14) \quad = \quad \sup_{\boldsymbol{\alpha},\, \boldsymbol{\lambda}} \quad \min_{M \succeq 0,\, \mathrm{tr}(M)=1} \quad -(n+1)\,\mathrm{tr}(MT(\boldsymbol{\alpha})) + \boldsymbol{\lambda}^\top (\mathbf{1} - (n+1)\boldsymbol{\delta}(M)) \qquad (15)$$

$$= \quad \sup_{\boldsymbol{\alpha},\, \boldsymbol{\lambda}} \quad \boldsymbol{\lambda}^\top \mathbf{1} - (n+1) \max_{M \succeq 0,\, \mathrm{tr}(M)=1} \mathrm{tr}\left[ M \left( T(\boldsymbol{\alpha}) + \Delta(\boldsymbol{\lambda}) \right) \right]. \qquad (16)$$

This relaxed formulation (16) is now amenable to efficient global optimization: The outer problem is jointly concave in $\boldsymbol{\alpha}$ and $\boldsymbol{\lambda}$, since it is a pointwise minimum of concave functions. The inner optimization with respect to $M$ can now be simplified by exploiting the well known result [21]:

$$\max_{M \succeq 0,\, \mathrm{tr}(M)=1} \mathrm{tr}\left[ M \left( T(\boldsymbol{\alpha}) + \Delta(\boldsymbol{\lambda}) \right) \right] = \max_{\|\boldsymbol{\nu}\|=1} \boldsymbol{\nu}^\top \left[ T(\boldsymbol{\alpha}) + \Delta(\boldsymbol{\lambda}) \right] \boldsymbol{\nu}. \qquad (17)$$

Therefore, given $\boldsymbol{\alpha}$ and $\boldsymbol{\lambda}$, the inner problem is solved by the maximum eigenvector of $T(\boldsymbol{\alpha}) + \Delta(\boldsymbol{\lambda})$.

**Optimization Procedure:** Given training data, an outer maximization can be executed jointly over $\boldsymbol{\alpha}$ and $\boldsymbol{\lambda}$ to maximize (16). This outer problem is concave in $\boldsymbol{\alpha}$ and $\boldsymbol{\lambda}$ hence no local maxima exist. Although the outer problem is not smooth, many effective methods exist for nonsmooth convex optimization [20, 31]. Each outer function evaluation (and subgradient calculation) requires the inner problem (17) to be solved. Fortunately, a simple power method [10] can be used to efficiently compute a maximum eigenvector solution to the inner problem by only performing matrix-vector multiplications on the individual factors of the two low rank matrices making up $T(\boldsymbol{\alpha})$, meaning the inner problem can be solved without ever forming a large $n \times n$ matrix $T(\boldsymbol{\alpha})$. That is, if $X$ is $n \times m$ each inner iteration requires at most $O(nm)$ computation.

**Solution Recovery:** At a solution, the values of (13)–(16) are equal, and all provide a lower bound on the original objective (6). Ideally, given a maximizer $\alpha^*$ for (14) one would recover a prediction model $\boldsymbol{\theta}$ via (12). However, (12) requires $\boldsymbol{\rho}$ to be acquired first, which could be obtained from a $\boldsymbol{\nu}$ that solves (10). Unfortunately, the relaxation step taken in (13) means that the solution to (14) (recovered from the $\boldsymbol{\nu}$ that solves (17)) does not necessarily solve (10): the inner solution $\boldsymbol{\nu}$ in (17) might not be unique. If it is unique, we immediately have the optimal solution to (10) hence an exact solution to the original problem (6). More typically, however, the maximum eigenvector is not unique at $(\boldsymbol{\alpha}^*, \boldsymbol{\lambda}^*)$, meaning that a gap has been introduced—this occurs if and only if the inner solution $M^*$ to (14) is not rank 1. In such cases we need to use a rounding procedure to recover an effective rank 1 approximation.

**Rounding Method:** Given the inner maximizer $(\boldsymbol{\alpha}^*, \boldsymbol{\lambda}^*)$ of (16) we do not need to explicitly construct the outer minimizer $M^*$. Instead, it suffices to construct a basis for $M^*$ by collecting the set of maximum eigenvectors $\tilde{V} = \{\tilde{\boldsymbol{\nu}}_1, ..., \tilde{\boldsymbol{\nu}}_k\}$ of $T(\boldsymbol{\alpha}^*) + \Delta(\boldsymbol{\lambda}^*)$ in (17) (note that $k$ is usually much smaller than $n+1$). A solution can then be indirectly obtained by solving a small semidefinite program to recover a $k \times k$ matrix $C^*$ that satisfies $C^* \succeq 0$ and $\boldsymbol{\delta}(\tilde{V}C^*\tilde{V}^\top) = \frac{1}{n+1}\mathbf{1}$. Note that $C^* = Q^*\Sigma^*Q^{*\top}$ for some orthonormal $Q^*$ and diagonal $\Sigma^*$ where $\sigma_j^* \geq 0$ and $\sum_{j=1}^{k} \sigma_j^* = 1$, hence $M^* = V^*\Sigma^*V^{*\top}$ such that $V^* = \{\boldsymbol{\nu}_1^*, ..., \boldsymbol{\nu}_k^*\} = \tilde{V}Q^*$. Given $V^*$ and $\Sigma^*$ a rounded solution for $\hat{\boldsymbol{\rho}}$ can be recovered simply by computing $\bar{\boldsymbol{\nu}}^* = \sum_{j=1}^{k} \sigma_j^* \boldsymbol{\nu}_j^*$ then setting $\hat{\boldsymbol{\rho}} = \frac{1}{2}\left(\mathbf{1} + \bar{\boldsymbol{\nu}}_{2:n+1}^* \sqrt{n+1}\right)$. From the constraints on $C^*$ it follows that $\frac{-1}{\sqrt{n+1}} \leq \bar{\nu}_j^* \leq \frac{1}{\sqrt{n+1}}$ hence $0 \leq \hat{\rho}_j \leq 1 \ \forall j$ (details in the supplement). Finally, instead of relying on (12) to recover the model parameters $\hat{\boldsymbol{\theta}}$ from $\hat{\boldsymbol{\rho}}$, we explicitly minimize the $\hat{\boldsymbol{\rho}}$-relaxed loss (7) given $\hat{\boldsymbol{\rho}}$ to recover $\hat{\boldsymbol{\theta}}$ via $\hat{\boldsymbol{\theta}} = \arg\min_{\boldsymbol{\theta}} R(\hat{\boldsymbol{\rho}}, \boldsymbol{\theta})$.

Although the rounding step has introduced an approximation, we establish that bounded outlier sensitivity can still be retained, even after the above relaxation and rounding processes, and demonstrate experimentally that the gap from optimality is generally not too large.

## 5 Bounding Outlier Sensitivity

Thus far we have proposed a robust training objective, provided an efficient convex relaxation that establishes a lower bound, and proposed a simple rounding method for recovering an approximate solution. The question remains as to whether the approximate solution retains bounded sensitivity to outliers (or to leverage points [23, §1.1]). Let $(\boldsymbol{\rho}^*, \boldsymbol{\theta}^*)$ denote the joint minimizer of (6) and let $(\hat{\boldsymbol{\rho}}, \hat{\boldsymbol{\theta}})$ denote the approximate solution obtained from the procedure above.

First, observe that an upper bound on the approximation error can be easily computed by subtracting the lower bound value obtained in (14)–(16) from $R(\hat{\boldsymbol{\rho}}, \hat{\boldsymbol{\theta}})$. Our experiments below show that reasonable gaps are obtained in this way. Nevertheless one would still like to guarantee that the gap stays bounded in the presence of arbitrary outliers and leverage points.

**Theorem 2** $\hat{R}(\hat{\boldsymbol{\rho}}, \boldsymbol{\alpha}^*) \leq 2R(\boldsymbol{\rho}^*, \boldsymbol{\theta}^*) \leq 2n$, where $\hat{R}(\hat{\boldsymbol{\rho}}, \boldsymbol{\alpha}^*)$ is the value of (10) at the rounded solution $\hat{\boldsymbol{\rho}}$. Furthermore, if the unclipped loss $\ell(y, \hat{y})$ is b-Lipschitz in $\hat{y}$ for $b < \infty$ and either $\mathbf{y}$ or $K$ remains bounded, then there exists a $c < \infty$ such that $R(\hat{\boldsymbol{\rho}}, \hat{\boldsymbol{\theta}}) \leq c$.

That is, the $\rho$-relaxed loss obtained by the rounded solution stays bounded in this case, even when accounting for the proposed relaxation and rounding procedure and data perturbation. The complete proof takes some space, however the key steps are to show that $-(n+1)\text{tr}(M^*T(\boldsymbol{\alpha}^*)) \leq R(\boldsymbol{\rho}^*, \boldsymbol{\theta}^*)$, and then use this to establish that $\hat{R}(\hat{\boldsymbol{\rho}}, \boldsymbol{\alpha}^*) \leq 2R(\boldsymbol{\rho}^*, \boldsymbol{\theta}^*)$ and $R(\hat{\boldsymbol{\rho}}, \hat{\boldsymbol{\theta}}) \leq c$, respectively (full details in the supplement). Thus, $(\hat{\boldsymbol{\rho}}, \hat{\boldsymbol{\theta}})$ will not chase outliers or leverage points arbitrarily in this situation. Note that the proposed method cannot be characterized by minimizing a fixed convex loss. That is, the tightest convex upper bound for any convex loss function is simply given by the function itself, which corresponds to setting $\rho_i = 1$ for every training example. By contrast, our approximation method does not choose a constant $\rho_i = 1$ for every training example, but instead *adaptively* chooses $\rho_i$ values, closer to 1 for inliers and closer to 0 for outliers. The resulting upper bound on the clipped loss (hence on the misclassification error in the margin loss case) is much tighter than that achieved by simply minimizing a convex loss. This outcome is demonstrated clearly in our experiments.

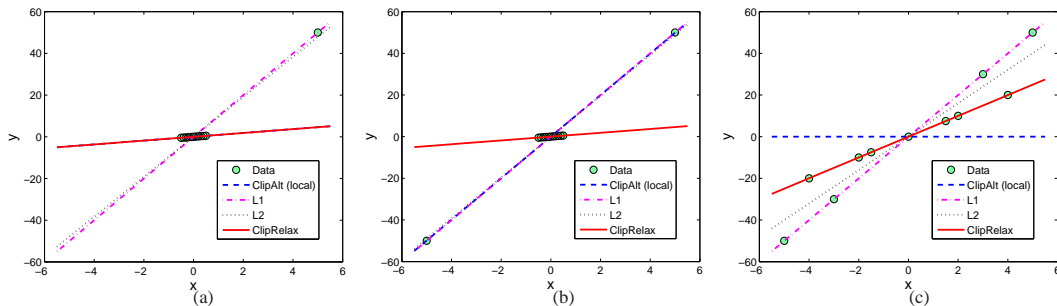

Figure 2: Comparison on three demonstration data sets.

|  | outlier probability | | |
| --- | --- | --- | --- |
| Loss | $p = 0.0$ | $p = 0.2$ | $p = 0.4$ |
| L2 | $2.53 \pm 0.0015$ | $25.11 \pm 13.78$ | $19.04 \pm 15.62$ |
| L1 | $2.53 \pm 0.0015$ | $26.52 \pm 16.09$ | $27.14 \pm 22.40$ |
| HuberM | $2.52 \pm 0.0015$ | $12.02 \pm 5.33$ | $12.30 \pm 5.87$ |
| GM (local) | $2.53 \pm 0.0015$ | $2.60 \pm 0.10$ | $2.62 \pm 0.09$ |
| ClipAlt (local) | $2.53 \pm 0.0019$ | $2.75 \pm 0.27$ | $2.81 \pm 0.27$ |
| ClipRelax | $2.53 \pm 0.0016$ | $2.68 \pm 0.12$ | $2.53 \pm 0.87$ |
| OptimGap | $1.65\% \pm 0.31\%$ | $0.10\% \pm 0.22\%$ | $0.70\% \pm 1.31\%$ |

Table 1: Synthetic experiment with $n = 200$, $m = 5$, and $t = 500$. Test error rates (RMSE) on clean data (average $\pm$ standard deviations) at different outlier probabilities $p$, 20 repeats. The bottom row shows the relative gap obtained between the $\rho$-relaxed loss of the rounded solution and the computed lower bound (16).

## 6 Experimental Results

In this section, we experimentally evaluate the preceding technical developments on synthetic and real data for both regression and classification.

**Regression:** We first illustrate the behavior of the various regression techniques by a simple demonstration. In Figure 2 (a) and (b), we generate a cluster of linearly related data $y = x$ in a small interval about the origin, then add outliers. In Figure 2 (c) the target linear model is mixed with another more dispersed model. We compare the behaviours of standard regression losses: least-squares (L2), $L_1$ (L1), the Huber minimax loss (HuberM) [13, 17], and the robust Geman and McClure loss (GM) [2]. To these we compare the proposed relaxed method (ClipRelax), along with an alternating minimizer of the clipped loss (ClipAlt). (In this problem the value of $\gamma$ has little effect, and is simply set to 0.1.) Figure 2 demonstrates that the three convex losses, L2, L1 and HuberM, are dominated by outliers. By contrast, ClipRelax successfully found the correct linear model in each case. Note that the robust GM loss finds two different minima, corresponding to that of L2 and ClipRelax respectively, hence it was not depicted in the plot. ClipAlt also gets trapped in local minima as expected: it finds the correct model in Figure 2 (a) but incorrect models in Figure 2 (b) and (c).

In our second synthetic regression experiment we consider larger problems. Here a target weight vector $\boldsymbol{\theta}$ is drawn from $N(\mathbf{0}, I)$, with inputs $X_{i:}$ sampled uniformly from $[0, 1]^m$, $m = 5$. The outputs $y_i$ are computed as $y_i = X_{i:}\boldsymbol{\theta} + \epsilon_i$, $\epsilon_i \sim N(0, \frac{1}{4})$. We then seed the data set with outliers by randomly re-sampling each $y_i$ (and $X_{i:}$) from $N(0, 10^5)$ and $N(0, 10^2)$ respectively, governed by an outlier probability $p$. Here 200 of the 700 examples are randomly chosen as the training set and the rest used for testing. We compare the same six methods: L2, L1, HuberM, GM, ClipAlt and ClipRelax. The regularization parameter $\gamma$ was set on a separate validation set. These experiments are repeated 20 times and average (Huber loss) test errors on clean data are reported (with standard deviations) in Table 1. Clearly, the outliers significantly affect the performance of least squares. In this case the proposed relaxation performs comparably to the the non-convex GM loss. Interestingly, this experiment shows that the relative gap between the $\rho$-robust loss obtained by the proposed method and the lower bound on the optimal $\rho$-robust loss (16) remains remarkably small, indicating our robust relaxation (almost) optimally minimizes the original non-convex clipped loss.

| Loss | Astronomy (1, 46, 46) | Cal-housing (8, 100, 1000) | Pumadyn (32, 500, 1000) |
|---|---|---|---|
| L2 | 2.484 | $804.5 \pm 892.5$ | $1.300 \times 10^5 \pm 68.29$ |
| L1 | 0.170 | $0.325 \pm 0.046$ | $5.133 \pm 0.056$ |
| HuberM | 0.149 | $0.306 \pm 0.050$ | $5.377 \pm 0.007$ |
| GM (local) | 0.166 | $0.329 \pm 0.048$ | $4.399 \pm 0.003$ |
| ClipAlt (local) | 0.176 | $0.329 \pm 0.048$ | $4.075 \pm 1 \times 10^{-6}$ |
| ClipRelax | 0.131 | $0.136 \pm 0.155$ | $4.075 \pm 1 \times 10^{-6}$ |

Table 2: Error rates (average root mean squared error, $\pm$ standard deviations) for the different regression estimators on various data sets. The values of $(m, n, tt)$ are indicated for each data set, where $m$ is the number of features, $n$ is the number of training examples, and $tt$ the number of testing samples.

| Loss | $p = 0.02$ | $p = 0.05$ | $p = 0.1$ | $p = 0.2$ |
|---|---|---|---|---|
| Logit | $4.88 \pm 6.17$ | $1.61 \pm 7.23$ | $17.67 \pm 4.00$ | $19.53 \pm 2.91$ |
| 1-tanh (local) | $0.91 \pm 1.93$ | $2.30 \pm 2.85$ | $6.49 \pm 4.32$ | $13.96 \pm 3.38$ |
| ClipAlt (local) | $0.46 \pm 0.64$ | $1.51 \pm 1.45$ | $4.27 \pm 2.57$ | $11.32 \pm 3.48$ |
| ClipRelax | $0.26 \pm 0.34$ | $0.78 \pm 0.78$ | $2.49 \pm 3.38$ | $10.10 \pm 8.21$ |

Table 3: Misclassification error rates on clean data (average error, $\pm$ standard deviations) on the Long-Servedio problem [16] with increasing noise levels $p$.

Finally, we investigated the behavior of the regression methods on a few real data sets. We chose three data sets: astronomy data containing outliers from [23], and two UCI data sets, seeding the the UCI data sets with outliers. Test results are reported on clean data to avoid skewing the reported results. For UCI data, outliers were added by resampling $X_{i:}$ and $y_i$ from $N(0, 1000)$, with 5% outliers. The regularization parameter $\gamma$ was chosen through 10-fold cross validation on the training set. Note that in real regression problems one needs to obtain an estimate for the the scale, given by the true standard deviation of the noise in the data. Here we estimated the scale using the *mean absolute deviation*, a robust approach commonly used in the robust statistics literature [17]. In Table 2, one can see that on both data sets, ClipRelax clearly outperformed the other methods. L2 is clearly skewed by the outliers. Unsurprisingly, the classical robust loss functions, L1 and HuberM, perform better than L2 in the presence of outliers, but not as well as ClipRelax.

**Classification:** We investigated the well known case study from [16] and compared the proposed method to logistic regression (i.e. the logit, or binomial deviance loss [12]) and the robust $1 - \tanh$ loss [19] in a classification context. Here 200 examples were drawn from the target distribution with label noise applied at various levels. The experiment was repeated 50 times to obtain average results and standard deviations. Table 3 shows the test error performance in clean data of the different methods. From these results one can conclude that ClipRelax is more robust than standard logit training. Training with logit loss is slightly better than the $\tanh$ loss algorithm in terms of training loss, but not very significantly. It is interesting to see that when the prediction error is measured on clean labels ClipRelax generalizes significantly better than the robust $1 - \tanh$ loss. This implies that the classification model produced by ClipRelax is closer to the true model despite of the presence of outliers, demonstrating that the proposed method can be robust in a simple classification context.

# 7    Conclusion

We have proposed a robust estimation method for regression and classification based on a notion of "loss-clipping". Although the method is not as fast as standard convex training, it is scalable to problems of moderate size. The key benefit is competitive (or better) estimation quality than the state-of-the-art in robust estimation, while ensuring provable robustness to outliers and computable bounds on the optimality gap. To the best of our knowledge these two properties have never been previously achieved simultaneously. It would be interesting to investigate whether the techniques developed can also be applied to other forms of robust estimators from the classical literature, including GM, MM, L, R and S estimators [11, 13, 17, 23]. Connections with algorithmic stability [3] and influence function based analysis [5, 6, 11] merit further investigation. Obtaining tighter bounds on approximation quality that would enable a proof of consistency also remains an important challenge.

## Footnotes

[1]All results in this paper extend to reproducing kernel Hilbert spaces via the representer theorem [24], but for clarity of presentation we will use an explicit feature representation $X$ even though it is not a requirement.

# References

[1] A. Banerjee, S. Merugu, I. Dhillon, and J. Ghosh. Clustering with Bregman divergences. *Journal of Machine Learning Research*, 6:1705–1749, 2005.

[2] M. Black and A. Rangarajan. On the unification of line processes, outlier rejection, and robust statistics with applications in early vision. *International Journal of Computer Vision*, 19(1):57–91, 1996.

[3] O. Bousquet and A. Elisseeff. Stability and generalization. *J. of Machine Learning Research*, 2, 2002.

[4] S. Boyd and L. Vandenberghe. *Convex Optimization*. Cambridge U. Press, 2004.

[5] A. Christmann and I. Steinwart. On robustness properties of convex risk minimization methods for pattern recognition. *Journal of Machine Learning Research*, 5:1007–1034, 2004.

[6] A. Christmann and I. Steinwart. Consistency and robustness of kernel-based regression in convex risk minimization. *Bernoulli*, 13(3):799–819, 2007.

[7] M. Collins, R. Schapire, and Y. Singer. Logistic regression, AdaBoost and Bregman distances. *Machine Learning*, 48, 2002.

[8] Y. Freund. A more robust boosting algorithm, 2009. arXiv.org:0905.2138.

[9] Y. Freund and R. Schapire. A decision-theoretic generalization of on-line learning and an application to boosting. *Journal of Computer and Systems Sciences*, 55(1):119–139, 1997.

[10] G. Golub and C. Van Loan. *Matrix Computations*. Johns Hopkins U. Press, 1996.

[11] F. Hampel, E. Ronchetti, P. Rousseeuw, and W. Stahel. *Robust Statistics: The Approach Based on Influence Functions*. Wiley, 1986.

[12] T. Hastie, R. Tibshirani, and J. Friedman. *Elements of Statistical Learning*. Springer, 2nd edition, 2009.

[13] P. Huber and E. Ronchetti. *Robust Statistics*. Wiley, 2nd edition, 2009.

[14] J. Kivinen and M. Warmuth. Relative loss bounds for multidimensional regression problems. *Machine Learning*, 45:301–329, 2001.

[15] N. Krause and Y. Singer. Leveraging the margin more carefully. In *Proceedings of the International Conference on Machine Learning (ICML)*, 2004.

[16] P. Long and R. Servedio. Random classification noise defeats all convex potential boosters. *Machine Learning*, 78:287–304, 2010.

[17] R. Maronna, R.D. Martin, and V. Yohai. *Robust Statistics: Theory and Methods*. Wiley, 2006.

[18] H. Masnadi-Shirazi and N. Vasconcelos. On the design of loss functions for classification: theory, robustness to outliers, and SavageBoost. In *Advances in Neural Information Processing Systems (NIPS)*, volume 21, pages 1049–1056, 2008.

[19] L. Mason, J. Baxter, P. Bartlett, and M. Frean. Functional gradient techniques for combining hypotheses. In *Advances in Large Margin Classifiers*. MIT Press, 2000.

[20] Y. Nesterov. *Introductory Lectures on Convex Optimization*. Kluwer, 1994.

[21] M. Overton and R. Womersley. Optimality conditions and duality theory for minimizing sums of the largest eigenvalues of symmetric matrices. *Mathematical Programming*, 62(2):321–357, 1993.

[22] R. Rockafellar. *Convex Analysis*. Princeton U. Press, 1970.

[23] P. Rousseeuw and A. Leroy. *Robust Regression and Outlier Detection*. Wiley, 1987.

[24] B. Schoelkopf and A. Smola. *Learning with Kernels*. MIT Press, 2002.

[25] X. Shen, G. Tseng, X. Zhang, and W.-H. Wong. On $\psi$-learning. *Journal of the American Statistical Association*, 98(463):724–734, 2003.

[26] C. Stewart. Robust parameter estimation in computer vision. *SIAM Review*, 41(3), 1999.

[27] Y. Wu and Y. Liu. Robust truncated hinge loss support vector machines. *Journal of the American Statistical Association*, 102(479):974–983, 2007.

[28] H. Xu, C. Caramanis, and S. Mannor. Robust regression and Lasso. In *Advances in Neural Information Processing Systems (NIPS)*, volume 21, pages 1801–1808, 2008.

[29] H. Xu, C. Caramanis, and S. Mannor. Robustness and regularization of support vector machines. *Journal of Machine Learning Research*, 10:1485–1510, 2009.

[30] L. Xu, K. Crammer, and D. Schuurmans. Robust support vector machine training via convex outlier ablation. In *Proceedings of the National Conference on Artificial Intelligence (AAAI)*, 2006.

[31] J. Yu, S. Vishwanathan, S. Günter, and N. Schraudolph. A quasi-Newton approach to nonsmooth convex optimization problems in machine learning. *J. of Machine Learning Research*, 11:1145–1200, 2010.

